# Lattice Regression

**Eric K. Garcia**
Department of Electrical Engineering
University of Washington
Seattle, WA 98195
garciaer@ee.washington.edu

**Maya R. Gupta**
Department of Electrical Engineering
University of Washington
Seattle, WA 98195
gupta@ee.washington.edu

## Abstract

We present a new empirical risk minimization framework for approximating functions from training samples for low-dimensional regression applications where a lattice (look-up table) is stored and interpolated at run-time for an efficient implementation. Rather than evaluating a fitted function at the lattice nodes without regard to the fact that samples will be interpolated, the proposed lattice regression approach estimates the lattice to minimize the interpolation error on the given training samples. Experiments show that lattice regression can reduce mean test error by as much as $25\%$ compared to Gaussian process regression (GPR) for digital color management of printers, an application for which linearly interpolating a look-up table is standard. Simulations confirm that lattice regression performs consistently better than the naive approach to learning the lattice. Surprisingly, in some cases the proposed method — although motivated by computational efficiency — performs better than directly applying GPR with no lattice at all.

## 1 Introduction

In high-throughput regression problems, the cost of evaluating test samples is just as important as the accuracy of the regression and most non-parametric regression techniques do not produce models that admit efficient implementation, particularly in hardware. For example, kernel-based methods such as Gaussian process regression [1] and support vector regression require kernel computations between each test sample and a subset of training examples, and local smoothing techniques such as weighted nearest neighbors [2] require a search for the nearest neighbors.

For functions with a known and bounded domain, a standard *efficient* approach to regression is to store a regular lattice of function values spanning the domain, then interpolate each test sample from the lattice vertices that surround it. Evaluating the lattice is independent of the size of any original training set, but exponential in the dimension of the input space making it best-suited to low-dimensional applications. In digital color management — where real-time performance often requires millions of evaluations every second — the interpolated look-up table (LUT) approach is the most popular implementation of the transformations needed to convert colors between devices, and has been standardized by the International Color Consortium (ICC) with a specification called an ICC profile [3].

For applications where one begins with training data and must learn the lattice, the standard approach is to first estimate a function that fits the training data, then evaluate the estimated function at the lattice points. However, this is suboptimal because the effect of interpolation from the lattice nodes is not considered when estimating the function. This begs the question: can we instead learn lattice outputs that accurately reproduce the training data upon interpolation?

Iterative post-processing solutions that update a given lattice to reduce the post-interpolation error have been proposed by researchers in geospatial analysis [4] and digital color management [5]. In

this paper, we propose a solution that we term *lattice regression*, that jointly estimates all of the lattice outputs by minimizing the regularized interpolation error on the training data. Experiments with randomly-generated functions, geospatial data, and two color management tasks show that lattice regression consistently reduces error over the standard approach of evaluating a fitted function at the lattice points, in some cases by as much as $25\%$. More surprisingly, the proposed method can perform better than evaluating test points by Gaussian process regression using no lattice at all.

## 2 Lattice Regression

The motivation behind the proposed lattice regression is to jointly choose outputs for lattice nodes that interpolate the training data accurately. The key to this estimation is that the linear interpolation operation can be directly inverted to solve for the node outputs that minimize the squared error of the training data. However, unless there is ample training data, the solution will not necessarily be unique. Also, to decrease estimation variance it may be beneficial to avoid fitting the training data exactly. For these reasons, we add two forms of regularization to the minimization of the interpolation error. In total, the proposed form of lattice regression trades off three terms: empirical risk, Laplacian regularization, and a global bias. We detail these terms in the following subsections.

### 2.1 Empirical Risk

We assume that our data is drawn from the bounded input space $\mathcal{D} \subset \mathbb{R}^d$ and the output space $\mathbb{R}^p$; collect the training inputs $x_i \in \mathcal{D}$ in the $d \times n$ matrix $X = \begin{bmatrix} x_1, \ldots, x_n \end{bmatrix}$ and the training outputs $y_i \in \mathbb{R}^p$ in the $p \times n$ matrix $Y = \begin{bmatrix} y_1, \ldots, y_n \end{bmatrix}$. Consider a lattice consisting of $m$ nodes where $m = \prod_{j=1}^d m_j$ and $m_j$ is the number of nodes along dimension $j$. Each node consists of an input-output pair $(a_i \in \mathbb{R}^d, b_i \in \mathbb{R}^p)$ and the inputs $\{a_i\}$ form a grid that contains $\mathcal{D}$ within its convex hull. Let $A$ be the $d \times m$ matrix $A = \begin{bmatrix} a_1, \ldots, a_m \end{bmatrix}$ and $B$ be the $p \times m$ matrix $B = \begin{bmatrix} b_1, \ldots, b_m \end{bmatrix}$.

For any $x \in \mathcal{D}$, there are $q = 2^d$ nodes in the lattice that form a cell (hyper-rectangle) containing $x$ from which an output will be interpolated; denote the indices of these nodes by $c_1(x), \ldots, c_q(x)$. For our purposes, we restrict the interpolation to be a linear combination $\{w_1(x), \ldots, w_q(x)\}$ of the surrounding node outputs $\{b_{c_1(x)}, \ldots, b_{c_q(x)}\}$, i.e. $\hat{f}(x) = \sum_i w_i(x) b_{c_i(x)}$. There are many interpolation methods that correspond to distinct weightings (for instance, in three dimensions: trilinear, pyramidal, or tetrahedral interpolation [6]). Additionally, one might consider a higher-order interpolation technique such as tricubic interpolation, which expands the linear weighting to the nodes directly adjacent to this cell. In our experiments we investigate only the case of $d$-linear interpolation (e.g. bilinear/trilinear interpolation) because it is arguably the most popular variant of linear interpolation, can be implemented efficiently, and has the theoretical support of being the maximum entropy solution to the underdetermined linear interpolation equations [7].

Given the weights $\{w_1(x), \ldots, w_q(x)\}$ corresponding to an interpolation of $x$, let $W(x)$ be the $m \times 1$ sparse vector with $c_j(x)$th entry $w_j(x)$ for $j = 1, \ldots, 2^d$ and zeros elsewhere. Further, for training inputs $\{x_1, \ldots, x_n\}$, let $W$ be the $m \times n$ matrix $W = \begin{bmatrix} W(x_1), \ldots, W(x_n) \end{bmatrix}$. The lattice outputs $B^*$ that minimize the total squared-$\ell_2$ distortion between the lattice-interpolated training outputs $BW$ and the given training outputs $Y$ are

$$B^* = \underset{B}{\arg\min} \; \mathbf{tr}\Big( \big( BW - Y \big) \big( BW - Y \big)^T \Big). \tag{1}$$

### 2.2 Laplacian Regularization

Alone, the empirical risk term is likely to pose an underdetermined problem and overfit to the training data. As a form of regularization, we propose to penalize the average squared difference of the output on adjacent lattice nodes using *Laplacian regularization*. A somewhat natural regularization of a function defined on a lattice, its inclusion guarantees[1] an unique solution to (1).

The graph Laplacian [8] of the lattice is fully defined by the $m \times m$ lattice adjacency matrix $E$ where $E_{ij} = 1$ for nodes directly adjacent to one another and 0 otherwise. Given $E$, a normalized version

of the Laplacian can be defined as $L = 2(\text{diag}(\mathbf{1}^T E) - E)/(\mathbf{1}^T E \mathbf{1})$, where $\mathbf{1}$ is the $m \times 1$ all-ones vector. The average squared error between adjacent lattice outputs can be compactly represented as

$$\mathbf{tr}(BLB^T) = \sum_{k=1}^{p} \left( \frac{1}{\sum_{ij} E_{ij}} \sum_{\{i,j \mid E_{ij}=1\}} (B_{ki} - B_{kj})^2 \right).$$

Thus, inclusion of this term penalizes first-order differences of the function at the scale of the lattice.

## 2.3   Global Bias

Alone, the Laplacian regularization of Section 2.2 rewards smooth transitions between adjacent lattice outputs but only enforces regularity at the resolution of the nodes, and there is no incentive in either the empirical risk or Laplacian regularization term to extrapolate the estimated function beyond the boundary of the cells that contain training samples. When the training data samples do not span all of the grid cells, the lattice node outputs reconstruct a clipped function. In order to endow the algorithm with an improved ability to extrapolate and regularize towards trends in the data, we also include a *global bias* term in the lattice regression optimization. The global bias term penalizes the divergence of lattice node outputs from some global function $\tilde{f} : \mathbb{R}^d \rightarrow \mathbb{R}^p$ that approximates the training data and this can be learned using any regression technique.

Given $\tilde{f}$, we bias the lattice regression nodes towards $\tilde{f}$'s predictions for the lattice nodes by minimizing the average squared deviation:

$$\frac{1}{m}\mathbf{tr}\Big( \big(B - \tilde{f}(A)\big)\big(B - \tilde{f}(A)\big)^T \Big).$$

We hypothesized that the lattice regression performance would be better if the $\tilde{f}$ was itself a good regression of the training data. Surprisingly, experiments comparing an accurate $\tilde{f}$, an inaccurate $\tilde{f}$, and no bias at all showed little difference in most cases (see Section 3 for details).

## 2.4   Lattice Regression Objective Function

Combined, the empirical risk minimization, Laplacian regularization, and global bias form the proposed lattice regression objective. In order to adapt an appropriate mixture of these terms, the regularization parameters $\alpha$ and $\gamma$ trade-off the first-order smoothness and the divergence from the bias function, relative to the empirical risk. The combined objective solves for the lattice node outputs $B^*$ that minimize

$$\underset{B}{\arg\min} \ \mathbf{tr}\Big(\frac{1}{n}\big(BW - Y\big)\big(BW - Y\big)^T + \alpha BLB^T + \frac{\gamma}{m}\big(B - \tilde{f}(A)\big)\big(B - \tilde{f}(A)\big)^T\Big),$$

which has the closed form solution

$$B^* = \left(\frac{1}{n}YW^T + \frac{\gamma}{m}\tilde{f}(A)\right)\left(\frac{1}{n}WW^T + \alpha L + \frac{\gamma}{m}I\right)^{-1}, \tag{2}$$

where $I$ is the $m \times m$ identity matrix.

Note that this is a joint optimization over all lattice nodes simultaneously. Since the $m \times m$ matrix that is inverted in (2) is sparse (it contains no more than $3^d$ nonzero entries per row[2]), (2) can be solved using sparse Cholesky factorization [9]. On a 64bit 2.6GHz processor using the Matlab command `mldivide`, we found that we could compute solutions for lattices that contained on the order of $10^4$ nodes (a standard size for digital color management profiling [6]) in $< 20$s using $< 1GB$ of memory but could not compute solutions for lattices that contained on the order of $10^5$ nodes.

# 3 Experiments

The effectiveness of the proposed method was analyzed with simulations on randomly-generated functions and tested on a real-data geospatial regression problem as well as two real-data color management tasks. For all experiments, we compared the proposed method to Gaussian process regression (GPR) applied directly to the final test points (no lattice), and to estimating test points by interpolating a lattice where the lattice nodes are learned by the same GPR. For the color management task, we also compared a state-of-the art regression method used previously for this application: local ridge regression using the enclosing $k$-NN neighborhood [10]. In all experiments we evaluated the performance of lattice regression using three different global biases: 1) an "accurate" bias $\tilde{f}$ was learned by GPR on the training samples; an "inaccurate" bias $\tilde{f}$ was learned by a global $d$-linear interpolation[3]; and 3) the no bias case, where the $\gamma$ term in (2) is fixed at zero.

To implement GPR, we used the MATLAB code provided by Rasmussen and Williams at `http://www.GaussianProcess.org/gpml`. The covariance function was set as the sum of a squared-exponential with an independent Gaussian noise contribution and all data were demeaned by the mean of the training outputs before applying GPR. The hyperparameters for GPR were set by maximizing the marginal likelihood of the training data (for details, see Rasmussen and Williams [1]). To mitigate the problem of choosing a poor local maxima, gradient descent was performed from 20 random starting log-hyperparameter values drawn uniformly from $[-10, 10]^3$ and the maximal solution was chosen. The parameters for all other algorithms were set by minimizing the 10-fold cross-validation error using the Nelder-Mead simplex method, bounded to values in the range $[\mathtt{1e^{-3}}, \mathtt{1e^3}]$. The starting point for this search was set at the default parameter setting for each algorithm: $\lambda = 1$ for local ridge regression[4] and $\alpha = 1, \gamma = 1$ for lattice regression. Experiments on the simulated dataset comparing this approach to the standard cross-validation over a grid of values $[\mathtt{1e^{-3}}, \mathtt{1e^{-2}}, \dots, \mathtt{1e^3}] \times [\mathtt{1e^{-3}}, \mathtt{1e^{-2}}, \dots, \mathtt{1e^3}]$ showed no difference in performance, and the former was nearly 50% faster.

## 3.1 Simulated Data

We analyzed the proposed method with simulations on randomly-generated piecewise-polynomial functions $f : \mathbb{R}^d \to \mathbb{R}$ formed from splines. These functions are smooth but have features that occur at different length-scales; two-dimensional examples are shown in Fig. 1. To construct each function, we first drew ten iid random points $\{s_i\}$ from the uniform distribution on $[0, 1]^d$, and ten iid random points $\{t_i\}$ from the uniform distribution on $[0, 1]$. Then for each of the $d$ dimensions we first fit a one-dimensional spline $\tilde{g}_k : \mathbb{R} \to \mathbb{R}$ to the pairs $\{((s_i)_k, t_i)\}$, where $(s_i)_k$ denotes the $k$th component of $s_i$. We then combined the $d$ one-dimensional splines to form the $d$-dimensional function $\tilde{g}(x) = \sum_{k=1}^d \tilde{g}_k((x)_k)$, which was then scaled and shifted to have range spanning $[0, 100]$:

$$f(x) = 100 \left( \frac{\tilde{g}(x) - \min_{z \in [0,1]^d} \tilde{g}(z)}{\max_{z \in [0,1]^d} \tilde{g}(z)} \right).$$

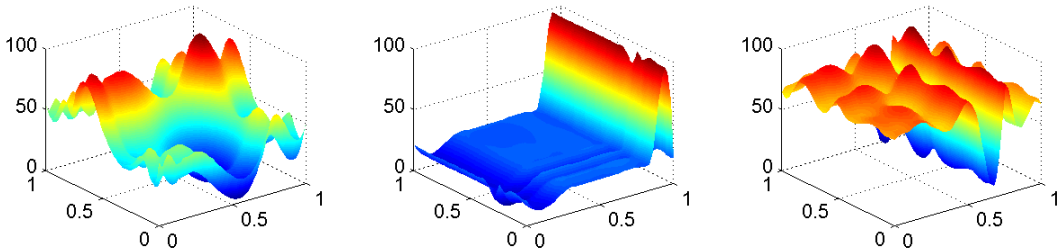

Figure 1: Example random piecewise-polynomial functions created by the sum of one-dimensional splines fit to ten uniformly drawn points in each dimension.

For input dimensions $d \in \{2, 3\}$, a set of 100 functions $\{f_1, \ldots, f_{100}\}$ were randomly generated as described above and a set of $n \in \{50, 1000\}$ randomly chosen training inputs $\{x_1, \ldots, x_n\}$ were fit by each regression method. A set of $m = 10,000$ randomly chosen test inputs $\{z_1, \ldots, z_m\}$ were used to evaluate the accuracy of each regression method in fitting these functions. For the $r$th randomly-generated function $f_r$, denote the estimate of the $j$th test sample by a regression method as $(\hat{y}_j)_r$. For each of the 100 functions and each regression method we computed the root mean-squared errors (RMSE) where the mean is over the $m = 10,000$ test samples:

$$e_r = \left( \frac{1}{m} \sum_{j=1}^{m} \left( f_r(z_j) - (\hat{y}_j)_r \right)^2 \right)^{1/2}.$$

The mean and statistical significance (as judged by a one-sided Wilcoxon with $p = 0.05$) of $\{e_r\}$ for $r = 1, \ldots, 100$ is shown in Fig. 2 for lattice resolutions of $5, 9$ and $17$ nodes per dimension.

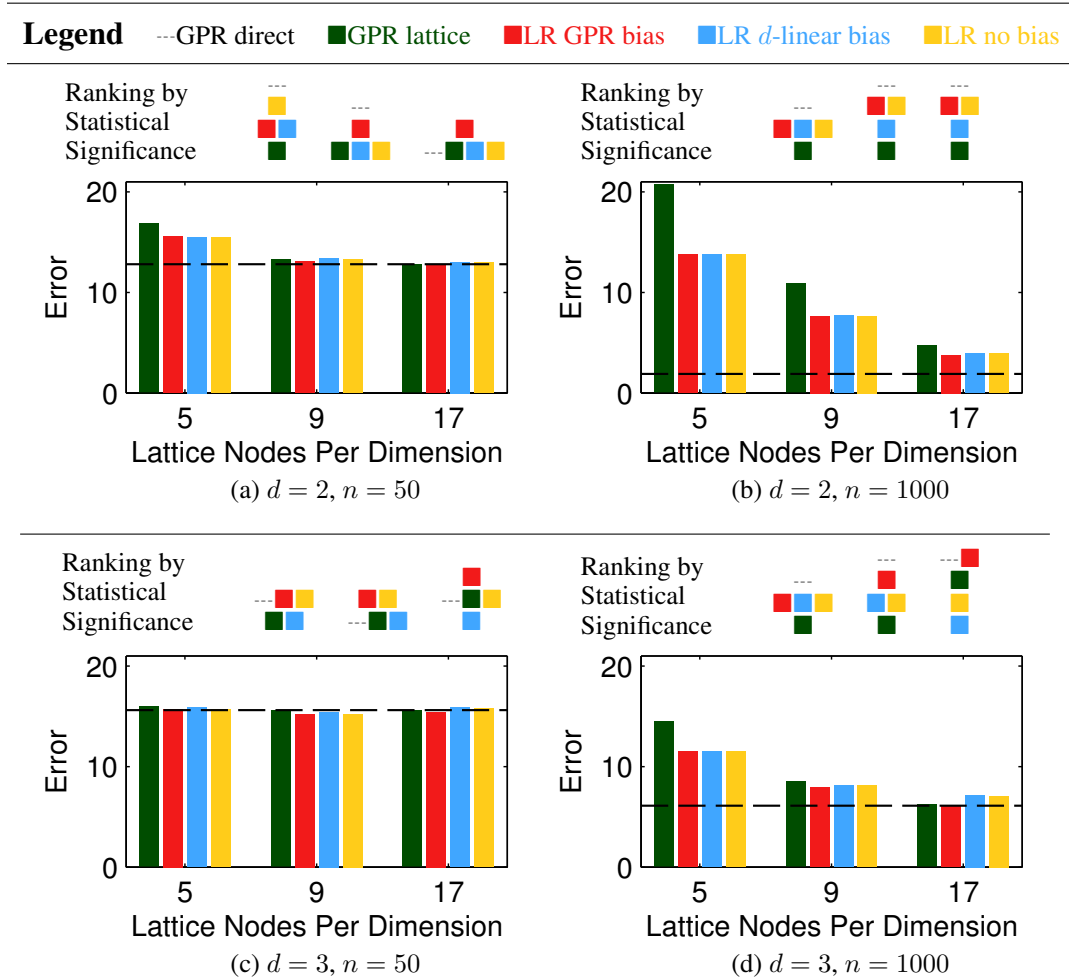

Figure 2: Shown is the average RMSE of the estimates given by each regression method on the simulated dataset. As summarized in the legend, shown is GPR applied directly to the test samples (dotted line) and the bars are (from left to right) GPR applied to the nodes of a lattice which is then used to interpolate the test samples, lattice regression with a GPR bias, lattice regression with a $d$-linear regression bias, and lattice regression with no bias. The statistical significance corresponding to each group is shown as a hierarchy above each plot: method A is shown as stacked above method B if A performed statistically significantly better than B.

In interpreting the results of Fig. 2, it is important to note that the statistical significance test compares the ordering of relative errors between each pair of methods *across the random functions*.

That is, it indicates whether one method consistently outperforms another in RMSE when fitting the randomly drawn functions.

Consistently across the random functions, and in all 12 experiments, lattice regression with a GPR bias performs better than applying GPR to the nodes of the lattice. At coarser lattice resolutions, the choice of bias function does not appear to be as important: in 7 of the 12 experiments (all at the low end of grid resolution) lattice regression using no bias does as well or better than that using a GPR bias.

Interestingly, in 3 of the 12 experiments, lattice regression with a GPR bias achieves statistically significantly lower errors (albeit by a marginal average amount) than applying GPR directly to the random functions. This surprising behavior is also demonstrated on the real-world datasets in the following sections and is likely due to large extrapolations made by GPR and in contrast, interpolation from the lattice regularizes the estimate which reduces the overall error in these cases.

### 3.2 Geospatial Interpolation

Interpolation from a lattice is a common representation for storing geospatial data (measurements tied to geographic coordinates) such as elevation, rainfall, forest cover, wind speed, etc. As a cursory investigation of the proposed technique in this domain, we tested it on the Spatial Interpolation Comparison 97 (SIC97) dataset [11] from the Journal of Geographic Information and Decision Analysis. This dataset is composed of 467 rainfall measurements made at distinct locations across Switzerland. Of these, 100 randomly chosen sites were designated as training to predict the rainfall at the remaining 367 sites. The RMSE of the predictions made by GPR and variants of the proposed method are presented in Fig 3. Additionally, the statistical significance (as judged by a one-sided Wilcoxon with $p = 0.05$) of the differences in squared error on the 367 test samples was computed for each pair of techniques. In contrast to the previous section in which significance was computed on the RMSE across 100 randomly drawn functions, significance in this section indicates that one technique produced consistently lower squared error across the individual test samples.

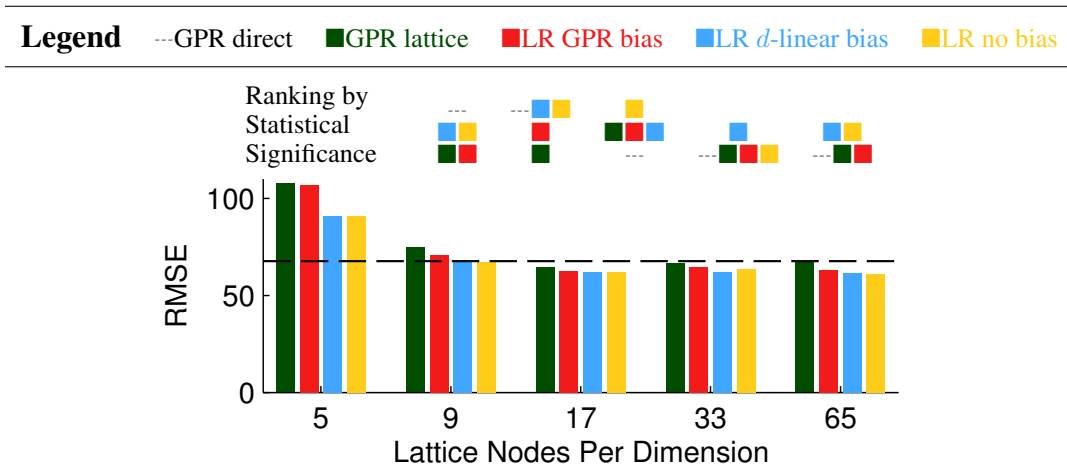

Figure 3: Shown is the RMSE of the estimates given by each method for the SIC97 test samples. The hierarchy of statistical significance is presented as in Fig. 2.

Compared with GPR applied to a lattice, lattice regression with a GPR bias again produces a lower RMSE on all five lattice resolutions. However, for four of the five lattice resolutions, there is no performance improvement as judged by the statistical significance of the individual test errors. In comparing the effectiveness of the bias term, we see that on four of five lattice resolutions, using no bias and using the $d$-linear bias produce consistently lower errors than both the GPR bias and GPR applied to a lattice.

Additionally, for finer lattice resolutions ($\geq 17$ nodes per dimension) lattice regression either outperforms or is not significantly worse than GPR applied directly to the test points. Inspection of the

maximal errors confirms the behavior posited in the previous section: that interpolation from the lattice imposes a helpful regularization. The range of values produced by applying GPR directly lies within $[1, 552]$ while those produced by lattice regression (regardless of bias) lie in the range $[3, 521]$; the actual values at the test samples lie in the range $[0, 517]$.

### 3.3   Color Management Experiments with Printers

Digital color management allows for a consistent representation of color information among diverse digital imaging devices such as cameras, displays, and printers; it is a necessary part of many professional imaging workflows and popular among semi-professionals as well. An important component of any color management system is the characterization of the mapping between the native color space of a device (RGB for many digital displays and consumer printers), and a device-independent space such as CIE L*a*b* — abbreviated herein as Lab — in which distance approximates perceptual notions of color dissimilarity [12].

For nonlinear devices such as printers, the color mapping is commonly estimated empirically by printing a page of color patches for a set of input RGB values and measuring the printed colors with a spectrophotometer. From these training pairs of (Lab, RGB) colors, one estimates the inverse mapping $f :$ Lab $\rightarrow$ RGB that specifies what RGB inputs to send to the printer in order to reproduce a desired Lab color. See Fig. 4 for an illustration of a color-managed system. Estimating $f$ is challenging for a number of reasons: 1) $f$ is often highly nonlinear; 2) although it can be expected to be smooth over regions of the colorspace, it is affected by changes in the underlying printing mechanisms [13] that can introduce discontinuities; and 3) device instabilities and measurement error introduce noise into the training data. Furthermore, millions of pixels must be processed in approximately real-time for every image without adding undue costs for hardware, which explains the popularity of using a lattice representation for color management in both hardware and software imaging systems.

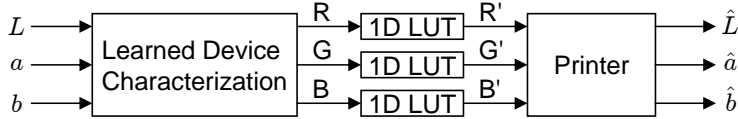

Figure 4: A color-managed printer system. For evaluation, errors are measured between the desired $(L, a, b)$ and the resulting $(\hat{L}, \hat{a}, \hat{b})$ for a given device characterization.

The proposed lattice regression was tested on an HP Photosmart D7260 ink jet printer and a Samsung CLP-300 laser printer. As a baseline, we compared to a state-of-the-art color regression technique used previously in this application [10]: local ridge regression (LRR) using the enclosing $k$-NN neighborhood. Training samples were created by printing the Gretag MacBeth TC9.18 RGB image, which has 918 color patches that span the RGB colorspace. We then measured the printed color patches with an X-Rite iSis spectrophotometer using D50 illuminant at a $2°$ observer angle and UV filter. As shown in Fig. 4 and as is standard practice for this application, the data for each printer is first gray-balanced using 1D calibration look-up-tables (1D LUTs) for each color channel (see [10, 13] for details). We use the same 1D LUTs for all the methods compared in the experiment and these were learned for each printer using direct GPR on the training data.

We tested each method's accuracy on reproducing 918 new randomly-chosen in-gamut[5] test Lab colors. The test errors for the regression methods the two printers are reported in Tables 1 and 2. As is common in color management, we report $\Delta E_{76}$ error, which is the Euclidean distance between the desired test Lab color and the Lab color that results from printing the estimated RGB output of the regression (see Fig. 4).

For both printers, the lattice regression methods performed best in terms of mean, median and 95 %-ile error. Additionally, according to a one-sided Wilcoxon test of statistical significance with

Table 1: Samsung CLP-300 laser printer

| | Euclidean Lab Error | | | |
| --- | --- | --- | --- | --- |
| | Mean | Median | 95 %-ile | Max |
| Local Ridge Regression (to fit lattice nodes) | 4.59 | 4.10 | 9.80 | 14.59 |
| GPR (direct) | 4.54 | 4.22 | 9.33 | 17.36 |
| GPR (to fit lattice nodes) | 4.54 | 4.17 | 9.62 | 15.95 |
| **Lattice Regression (GPR bias)** | **4.31** | **3.95** | **9.08** | **15.11** |
| **Lattice Regression (Trilinear bias)** | **4.14** | **3.75** | **8.39** | **15.59** |
| **Lattice Regression (no bias)** | **4.08** | **3.72** | **8.00** | **17.45** |

Table 2: HP Photosmart D7260 inkjet printer

| | Euclidean Lab Error | | | |
| --- | --- | --- | --- | --- |
| | Mean | Median | 95 %-ile | Max |
| Local Ridge Regression (to fit lattice nodes) | 3.34 | 2.84 | 7.70 | 14.77 |
| GPR (direct) | 2.79 | 2.45 | 6.36 | 11.08 |
| GPR (to fit lattice nodes) | 2.76 | 2.36 | 6.36 | 11.79 |
| Lattice Regression (GPR bias) | 2.53 | 2.17 | 5.96 | 10.25 |
| Lattice Regression (Trilinear bias) | 2.34 | 1.84 | 5.89 | 12.48 |
| **Lattice Regression (no bias)** | **2.07** | **1.75** | **4.89** | **10.51** |

The bold face indicates that the individual errors are statistically significantly lower than the others as judged by a one-sided Wilcoxon significance test (p=0.05). Multiple bold lines indicate that there was no statistically significant difference in the bolded errors.

$p = 0.05$, all of the lattice regressions (regardless of the choice of bias) were statistically significantly better than the other methods for both printers; on the Samsung, there was no significant difference between the choice of bias, and on the HP using the using no bias produced consistently lower errors. These results are surprising for three reasons. First, the two printers have rather different nonlinearities because the underlying physical mechanisms differ substantially (one is a laser printer and the other is an inkjet printer), so it is a nod towards the generality of the lattice regression that it performs best in both cases. Second, the lattice is used for computationally efficiency, and we were surprised to see it perform better than directly estimating the test samples using the function estimated with GPR directly (no lattice). Third, we hypothesized (incorrectly) that better performance would result from using the more accurate global bias term formed by GPR than using the very coarse fit provided by the global trilinear bias or no bias at all.

## 4    Conclusions

In this paper we noted that low-dimensional functions can be efficiently implemented as interpolation from a regular lattice and we argued that an optimal approach to learning this structure from data should take into account the effect of this interpolation. We showed that, in fact, one can directly estimate the lattice nodes to minimize the empirical interpolated training error and added two regularization terms to attain smoothness and extrapolation. It should be noted that, in the experiments, extrapolation beyond the training data was not directly tested: test samples for the simulated and real-data experiments were drawn mainly from within the interior of the training data.

Real-data experiments showed that mean error on a practical digital color management problem could be reduced by $25\%$ using the proposed lattice regression, and that the improvement was statistically significant. Simulations also showed that lattice regression was statistically significantly better than the standard approach of first fitting a function then evaluating it at the lattice points. Surprisingly, although the lattice architecture is motivated by computational efficiency, both our simulated and real-data experiments showed that the proposed lattice regression can work better than state-of-the-art regression of test samples without a lattice.

## Footnotes

[1]For large enough values of the mixing parameter $\alpha$.

[2]For a given row, the only possible non-zero entries of $WW^T$ correspond to nodes that are adjacent in one or more dimensions and these non-zero entries overlap with those of $L$.

[3]We considered the very coarse $m = 2^d$ lattice formed by the corner vertices of the original lattice and solved (1) for this one-cell lattice, using the result to interpolate the full set of lattice nodes, forming $\tilde{f}(A)$.

[4]Note that no locality parameter is needed for this local ridge regression as the neighborhood size is automatically determined by enclosing $k$-NN [10].

[5] We drew 918 samples iid uniformly over the RGB cube, printed these, and measured the resulting Lab values; these Lab values were used as test samples. This is a standard approach to assuring that the test samples are Lab colors that are in the achievable color gamut of the printer [10].

# References

[1] C. E. Rasmussen and C. K. I. Williams, *Gaussian Processes for Machine Learning (Adaptive Computation and Machine Learning)*, The MIT Press, 2005.

[2] T. Hastie, R. Tibshirani, and J. Friedman, *The Elements of Statistical Learning*, Springer-Verlag, New York, 2001.

[3] D. Wallner, *Building ICC Profiles - the Mechanics and Engineering*, chapter 4: ICC Profile Processing Models, pp. 150–167, International Color Consortium, 2000.

[4] W. R. Tobler, "Lattice tuning," *Geographical Analysis*, vol. 11, no. 1, pp. 36–44, 1979.

[5] R. Bala, "Iterative technique for refining color correction look-up tables," United States Patent 5,649,072, 1997.

[6] R. Bala and R. V. Klassen, *Digital Color Handbook*, chapter 11: Efficient Color Transformation Implementation, CRC Press, 2003.

[7] M. R. Gupta, R. M. Gray, and R. A. Olshen, "Nonparametric supervised learning by linear interpolation with maximum entropy," *IEEE Trans. on Pattern Analysis and Machine Intelligence (PAMI)*, vol. 28, no. 5, pp. 766–781, 2006.

[8] F. Chung, *Spectral Graph Theory*, Number 92 in Regional Conference Series in Mathematics. American Mathematical Society, 1997.

[9] T. A. Davis, *Direct Methods for Sparse Linear Systems*, SIAM, Philadelphia, September 2006.

[10] M. R. Gupta, E. K. Garcia, and E. M. Chin, "Adaptive local linear regression with application to printer color management," *IEEE Trans. on Image Processing*, vol. 17, no. 6, pp. 936–945, 2008.

[11] G. Dubois, "Spatial interpolation comparison 1997: Foreword and introduction," *Special Issue of the Journal of Geographic Information and Descision Analysis*, vol. 2, pp. 1–10, 1998.

[12] G. Sharma, *Digital Color Handbook*, chapter 1: Color Fundamentals for Digital Imaging, pp. 1–114, CRC Press, 2003.

[13] R. Bala, *Digital Color Handbook*, chapter 5: Device Characterization, pp. 269–384, CRC Press, 2003.

